# Learning Sparse Topographic Representations with Products of Student-t Distributions

**Max Welling and Geoffrey Hinton**
Department of Computer Science
University of Toronto
10 King's College Road
Toronto, M5S 3G5 Canada
{*welling,hinton*}*@cs.toronto.edu*

**Simon Osindero**
Gatsby Unit
University College London
17 Queen Square
London WC1N 3AR, UK
*simon@gatsby.ucl.ac.uk*

## Abstract

We propose a model for natural images in which the probability of an image is proportional to the product of the probabilities of some filter outputs. We encourage the system to find sparse features by using a Student-t distribution to model each filter output. If the t-distribution is used to model the combined outputs of sets of neurally adjacent filters, the system learns a topographic map in which the orientation, spatial frequency and location of the filters change smoothly across the map. Even though maximum likelihood learning is intractable in our model, the product form allows a relatively efficient learning procedure that works well even for highly overcomplete sets of filters. Once the model has been learned it can be used as a prior to derive the "iterated Wiener filter" for the purpose of denoising images.

## 1 Introduction

Historically, two different classes of statistical model have been used for natural images. "Energy-based" models assign to each image a global energy, $E$, that is the sum of a number of local contributions and they define the probability of an image to be proportional to $\exp(-E)$. This class of models includes Markov Random Fields where combinations of nearby pixel values contribute local energies, Boltzmann Machines in which binary pixels are augmented with binary hidden variables that learn to model higher-order statistical interactions and Maximum Entropy methods which learn the appropriate magnitudes for the energy contributions of heuristically derived features [5] [9]. It is difficult to perform maximum likelihood fitting on most energy-based models because of the normalization term (the partition function) that is required to convert $\exp(-E)$ to a probability. The normalization term is a sum over all *possible* images and its derivative w.r.t. the parameters is required for maximum likelihood fitting. The usual approach is to approximate this derivative by using Markov Chain Monte Carlo (MCMC) to sample from the model, but the large number of iterations required to reach equilibrium makes learning very slow.

The other class of model uses a "causal" directed acyclic graph in which the lowest level nodes correspond to pixels and the probability distribution at a node (in the absence of any observations) depends only on its parents. When the graph is singly or very sparsely

connected there are efficient algorithms for maximum likelihood fitting but if nodes have many parents, it is hard to perform maximum likelihood fitting because this requires the intractable posterior distribution over non-leaf nodes given the pixel values.

There is much debate about which class of model is the most appropriate for natural images. Is a particular image best characterized by the states of some hidden variables in a causal generative model? Or is it best characterized by its peculiarities *i.e.* by saying which of a very large set of normally satisfied constraints are violated? In this paper we treat violations of constraints as contributions to a global energy and we show how to learn a large set of constraints each of which is normally satisfied fairly accurately but occasionally violated by a lot. The ability to learn efficiently without ever having to generate equilibrium samples from the model and without having to confront the intractable partition function removes a major obstacle to the use of energy-based models.

## 2 The Product of Student-t Model

Products of Experts (PoE) are a restricted class of energy-based model [1]. The distribution represented by a PoE is simply the normalized product of all the distributions represented by the individual "experts":

$$P(\mathbf{x}) = \frac{1}{Z} \prod_{i=1}^{M} f_i(\mathbf{x}|\theta_i) \tag{1}$$

where $f_i(\mathbf{x}|\theta_i)$ are un-normalized experts and $Z$ denotes the overall normalization constant. In the product of Student-t (PoT) model, un-normalized experts have the following form,

$$f_i(\mathbf{x}) = \frac{1}{(1 + \frac{1}{2}(\mathbf{J}_i^T \mathbf{x})^2)^{\alpha_i}} \qquad \alpha_i > 0 \tag{2}$$

where $\mathbf{J}_i$ is called a *filter* and is the $i$-th column in the filter-matrix $\mathbf{J}$. When properly normalized, this represents a Student-t distribution over the filtered random variable $z_i = \mathbf{J}_i^T \mathbf{x}$. An important feature of the Student-t distribution is its heavy tails, which makes it a suitable candidate for modelling constraints of the kind that are found in images.

Defining $P(x) = \exp(-E(x))/Z$, the energy of the PoT model becomes

$$E(x) = \sum_{i=1}^{M} \alpha_i \log(1 + \frac{1}{2}(\mathbf{J}_i^T \mathbf{x})^2) \tag{3}$$

Viewed this way, the model takes the form of a maximum entropy distribution with weights $\alpha_i$ on real-valued "features" of the image. Unlike previous maximum entropy models, however, we can fit both the weights and the features at the same time.

When the number of input dimensions is equal to the number of experts, the normally intractable partition function becomes a determinant and the PoT model becomes equivalent to a noiseless ICA model with Student-t prior distributions [2]. In that case the rows of the inverse filters $\mathbf{A} = \mathbf{J}^{-1}$ will represent independent directions in input space. So noiseless ICA can be viewed as an energy-based model even though it is usually interpeted as a causal generative model in which the posterior over the hidden variables collapses to a point. However, when we consider more experts than input dimensions (i.e. an overcomplete representation), the energy-based view and the causal generative view lead to different generalizations of ICA. The natural causal generalization retains the independence of the hidden variables in the prior by assuming independent sources. In contrast, the PoT model simply multiplies together more experts than input dimensions and re-normalizes to get the total probability.

# 3   Training the PoT Model with Contrastive Divergence

When training energy-based models we need to shape the energy function so that observed images have low energy *and* empty regions in the space of all possible images have high energy. The maximum likelihood learning rule is given by,

$$\delta\theta \propto - \left\langle \frac{\partial E}{\partial \theta} \right\rangle_{\mathbf{data}} + \left\langle \frac{\partial E}{\partial \theta} \right\rangle_{\mathbf{equilibrium}} \tag{4}$$

It is the second term which causes learning to be slow and noisy because it is usually necessary to use MCMC to compute the average over the equilibrium distribution. A much more efficient way to fit the model is to use the data distribution itself to initialize a Markov Chain which then starts moving towards the model's equilibrium distribution. After just a few steps, we observe how the chain is diverging from the data and adjust the parameters to counteract this divergence. This is done by lowering the energy of the data and raising the energy of the "confabulations" produced by a few steps of MCMC.

$$\delta\theta \propto - \left\langle \frac{\partial E}{\partial \theta} \right\rangle_{\mathbf{data}} + \left\langle \frac{\partial E}{\partial \theta} \right\rangle_{\mathbf{k-step\ samples}} \tag{5}$$

It can be shown that the above update rule approximately minimizes a new objective function called the contrastive divergence [1].

As it stands the learning rule will be inefficient if the Markov Chain mixes slowly because the two terms in equation 5 will almost cancel each other out. To speed up learning we need a Markov chain that mixes rapidly so that the confabulations will be some distance away from the data. Rapid mixing can be achieved by alternately Gibbs sampling a set of hidden variables given the random variables under consideration and vice versa. Fortunately, the PoT model can be equipped with a number of hidden random variables equal to the number of experts as follows,

$$P(\mathbf{x}, \mathbf{u}) \propto e^{- \sum_{i=1}^{M} \left[ u_i \left(1 + \frac{1}{2}(\mathbf{J}_i^T \mathbf{x})^2\right) + (1 - \alpha_i) \log u_i \right]} \tag{6}$$

Integrating over the $\mathbf{u}$ variables results in the density of the PoT model, i.e. eqns. (1) and (2). Moreover, the conditional distributions are easy to identify and sample from, namely

$$P(\mathbf{u}|\mathbf{x}) \;=\; \prod_{i=1}^{M} \mathcal{G}_{u_i} \left[ \alpha_i \;;\; 1 + \frac{1}{2}(\mathbf{J}_i^T \mathbf{x})^2 \right] \tag{7}$$

$$P(\mathbf{x}|\mathbf{u}) \;=\; \mathcal{N}_{\mathbf{x}} \left[ 0 \;;\; (\mathbf{J}\mathbf{U}\mathbf{J}^T)^{-1} \right] \qquad \mathbf{U} = \mathbf{Diag}[u_i] \tag{8}$$

where $\mathcal{G}$ denotes a Gamma distribution and $\mathcal{N}$ a normal distribution. From (8) we see that the variables $\mathbf{u}$ can be interpreted as *precision* variables in the transformed space $\mathbf{z} = \mathbf{J}^T \mathbf{x}$. In this respect our model resembles a "Gaussian scale mixture" (GSM) [8] which also multiplies a positive scaling variable with a normal variate. But GSM is a causal model while PoT is energy-based.

The (in)dependency relations between the variables in a PoT model are depicted graphically in figure (1a,b). The hidden variables are independent given $\mathbf{x}$, which allows them to be Gibbs-sampled in parallel. This resembles the way in which brief Gibbs sampling is used to fit binary "Restricted Boltzmann Machines" [1].

To learn the parameters of the PoT model we thus propose to iterate the following steps:

1) Sample $\hat{\mathbf{u}}_n$ given the data $\mathbf{d}_n$ for every data-vector according to the Gamma-distribution (7).

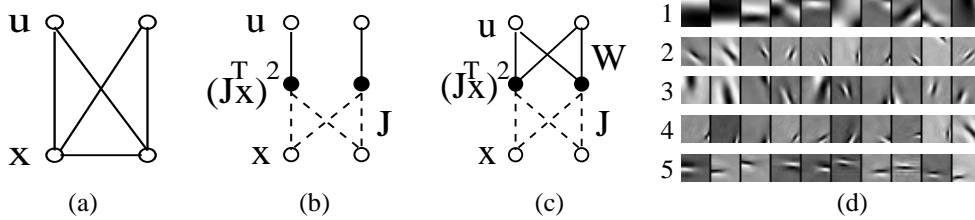

Figure 1: (a)- Undirected graph for the PoT model. (b)-Expanded graph where the deterministic relation (dashed lines) between the random variable $\mathbf{x}$ and the activities of the filters $(\mathbf{J}^T\mathbf{x})^2$ is made explicit. (c)-Graph for the PoT model including weights $\mathbf{W}$. (d)-Filters with large (decreasing from left to right) weights into a particular top level unit $u_i$. Top level units have learned to connect to filters similar in frequency, location and orientation.

2) Sample *reconstructions* of the data $\hat{\mathbf{x}}_n$ given the sampled values of $\hat{\mathbf{u}}_n$ for every data-vector according to the Normal distribution (8).

3) Update the parameters according to (5) where the "k-step samples" are now given by the reconstructions $\hat{\mathbf{x}}_n$, the energy is given by (3), and the parameters are given by $\theta = \{\mathbf{J}, \boldsymbol{\alpha}\}$.

## 4 Overcomplete Representations

The above learning rules are still valid for overcomplete representations. However, step-2 of the learning algorithm is much more efficient when the inverse of the filter matrix $\mathbf{J}$ exists. In that case we simply draw $N$ standard normal random numbers (with $N$ the number of data-vectors) and multiply each of them with $\mathbf{J}^{-T}\mathbf{U}_n^{-\frac{1}{2}}$. This is efficient because the data dependent matrix $\mathbf{U}_n^{-\frac{1}{2}}$ is diagonal while the costly inverse $\mathbf{J}^{-T}$ is data independent. In contrast, for the overcomplete case we ought to perform a Cholesky factorization on $\mathbf{J}\mathbf{U}_n\mathbf{J}^T$ for each data-vector separately. We have, however, obtained good results by proceeding as in the complete case and replacing the inverse of the filter matrix with its pseudo-inverse.

From experiments we have also found that in the overcomplete case we should fix the $L_2$-norm of the filters, $\mathbf{J}_i^T\mathbf{J}_i = 1 \; \forall i$, in order to prevent some of them from decaying to zero. This operation is done after every step of learning. Since controlling the norm removes the ability of the experts to adapt to scale it is necessary to whiten the data first.

### 4.1 Experiment: Overcomplete Representations for Natural Images

We randomly generated $50,000$ patches of $10 \times 10$ pixels from images of natural scenes[1]. The patches were centered and sphered using PCA and the DC component (eigen-vector with largest variance) was removed. The algorithm for overcomplete representations using the pseudo-inverse was used to train $50,176$ experts, i.e. a representation that is more than $500$ times overcomplete. We fixed the weights to have $\alpha_i = 1$ and the the filters to have a $L_2$-norm of 1. A small weight decay term and a momentum term were included in the gradient updates of the filters. The learning rate was set so that initially the change in the filters was approximately $0.001$. In figure (2a) we show a small subset of the inverse-filters given by the pseudo-inverse of $\mathbf{J}^T\mathbf{M_{PCA}}$, where $\mathbf{M_{PCA}}$ is the $100 \times 99$ matrix used for sphering the data.

## 5  Topographically Ordered Features

In [6] it was shown that linear filtering of natural images is not enough to remove all higher order dependencies. In particular, it was argued that there are residual dependencies among the activities $z_i^2 = (\mathbf{J}_i^T \mathbf{x})^2$ of the filtered inputs. It is therefore desirable to model those dependencies within the PoT model. By inspection of figure (1b) we note that these dependencies can be modelled through a non-negative weight matrix $W_{ij} \geq 0$, which connects the hidden variables $u_i$ with the activities $(\mathbf{J}_j^T \mathbf{x})^2$. The resultant model is depicted in figure (1c). Depending on how many nonzero weights $W_{ij}$ emanate from a hidden unit $u_i$ (say $d_i$), each expert now occupies $d_i$ input dimensions instead of just one. The expressions for these richer experts can be obtained from (2) by replacing, $(\mathbf{J}_i^T \mathbf{x})^2 \rightarrow \sum_j W_{ij}(\mathbf{J}_j^T \mathbf{x})^2$. We have found that learning is assisted by fixing the $L_1$-norm of the weights ($\sum_j W_{ij} = 1 \ \forall i$). Moreover, we have found that the sparsity of the weights $\mathbf{W}$ can be controlled by the following generalization of the experts,

$$f_i(\mathbf{x}) = \frac{1}{(1 + \frac{1}{2} \sum_{j=1}^{d_i} W_{ij}|\mathbf{J}_j^T \mathbf{x}|^\beta)^{\alpha_i}} \qquad \alpha_i, \beta > 0, \ W_{ij} \geq 0 \tag{9}$$

The larger the value for $\beta$ the sparser the distribution of $\mathbf{W}$ values.

Joint and conditional distributions over hidden variables are obtained through similar replacements in eqn. (6) and (7) respectively. Sampling the reconstructions given the states of the hidden variables proceeds by first sampling from $M$ independent generalized Laplace distributions $y_i \sim C_i \exp(-\tau_i|y_i|^\beta)$ with precision parameters $\boldsymbol{\tau} = \frac{1}{2}\mathbf{W}^T\mathbf{u}$ which are subsequently transformed into $\hat{\mathbf{x}} = \mathbf{J}^{-T}\mathbf{y}$. Learning in this model therefore proceeds with only minor modifications to the algorithm described in the previous section.

When we learn the weight matrix $W_{ij}$ from image data we find that a particular hidden variable $u_i$ develops weights to the activities of filters similar in frequency, location and orientation. The $\mathbf{u}$ variables therefore integrate information from these filters and as a result develop certain invariances that resemble the behavior of complex cells. A similar approach was studied in [4] using a related causal model[2] in which a number of scale variables generate correlated variances for conditionally Gaussian experts. This results in topography when the scale-generating variables are non-adaptive and connect to a local neighborhood of filters only.

We will now argue that fixed local weights $\mathbf{W}$ also give rise to topography in the PoT model. The reason is that averaging the squares of randomly chosen filter outputs (eqn.9) produces an approximately Gaussian distribution which is a poor fit to the heavy-tailed experts. However, this "smoothing effect" may be largely avoided by averaging squared filter outputs that are highly correlated (i.e. ones that are similar in location, frequency and orientation). Since the averaging is local, this results in a topographic layout of the filters.

### 5.1  Experiment: Topographic Representations for Natural Images

For this experiment we collected $50,000$ image patches of size $16 \times 16$ pixels in the same way as described in section (4.1). The image data were sphered and reduced to 169 dimensions by removing 86 low variance and 1 high variance (DC) direction. We learned an overcomplete representation with 400 experts which were organized on a square $20 \times 20$ grid. Each expert connects with a fixed weight of $W_{ij} = 1/9$ to itself and all its 8 neighbors, where periodic boundary conditions were imposed for the experts on the boundary.

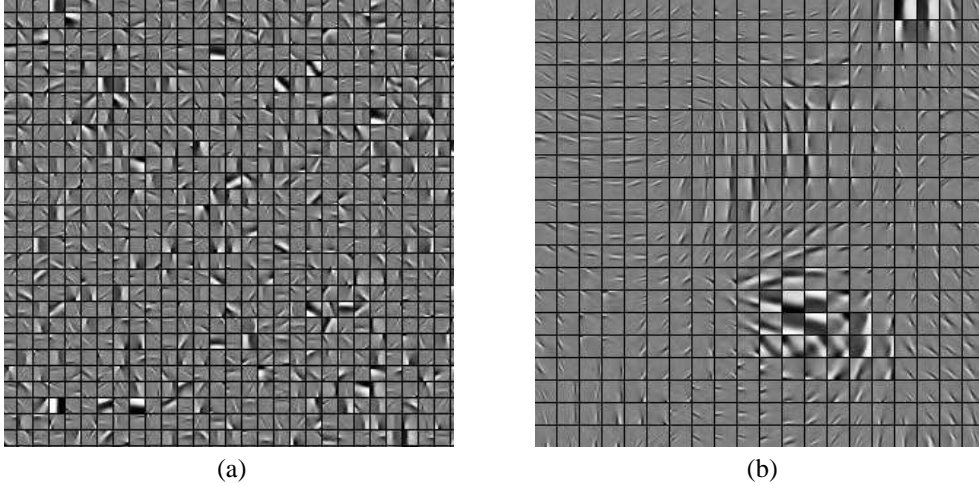

<div align="center">(a)                           (b)</div>

Figure 2: (a)-Small subset of the $50,176$ learned filters from a $500$ times overcomplete representation for natural image patches. (b)-Topographically ordered filters. The weights were fixed and connect to neighbors only, using periodic boundary conditions. Neighboring filters have learned to be similar in frequency, location and orientation. One can observe a pinwheel structure to the left of the low frequency cluster.

We adapted the filters $\mathbf{J}$ ($L_2$-norm 1) and used fixed values for $\alpha = 1$ and $\beta = 2$. The resulting inverse-filters are shown in figure (2b). We note that the weights $\mathbf{W}$ have enforced a topographic ordering on the experts, where location, scale and frequency of the Gabor-like filters all change smoothly across the map.

In another experiment we used the same data to train a complete representation of 169 experts where we learned the weights $\mathbf{W}$ ($L_1$-norm 1), $\alpha$ and the filters $\mathbf{J}$ (unconstrained), but with a fixed value of $\beta = 1$. The weights $\mathbf{W}$ and $\alpha$ were kept positive by adapting their logarithm. Since the weights $\mathbf{W}$ can now connect to any other expert we do not expect topography. To study whether the weights $\mathbf{W}$ were modelling the dependencies between the energies of the filter outputs $(\mathbf{J}_i^T \mathbf{x})^2$ we ordered the filters for each complex cell $u_i$ according to the strength of the weights connecting to it. For a representative subset of the complex cells $\mathbf{u}$, we show the 10 filters with the strongest connections to that cell in figure (1d). Since the cells connect to similar filters we may conclude that the weights $\mathbf{W}$ are indeed learning the dependencies between the activities of the filter outputs.

## 6  Denoising Images: The Iterated Wiener Filter

If the PoT model provides an accurate description of the statistics of natural image data it ought to be a good prior for cleaning up noisy images. In the following we will apply this idea to denoise images contaminated with Gaussian pixel noise. We follow the standard Bayesian approach which states that the optimal estimate of the original image is given by the maximum a posteriori (MAP) estimate of $P(\mathbf{x}|\mathbf{y})$, where $\mathbf{y}$ denotes the noisy image. For the PoT model this reduces to,

$$\mathbf{x}^{\mathbf{MAP}} = \mathbf{argmin}_{\mathbf{x}} \left[ \frac{1}{2}(\mathbf{y} - \mathbf{x})^T \mathbf{\Sigma}^{-1}(\mathbf{y} - \mathbf{x}) + \sum_i \alpha_i \log \left( 1 + \frac{1}{2} \sum_j W_{ij}(\mathbf{J}_j^T \mathbf{x})^2 \right) \right]$$

(10)

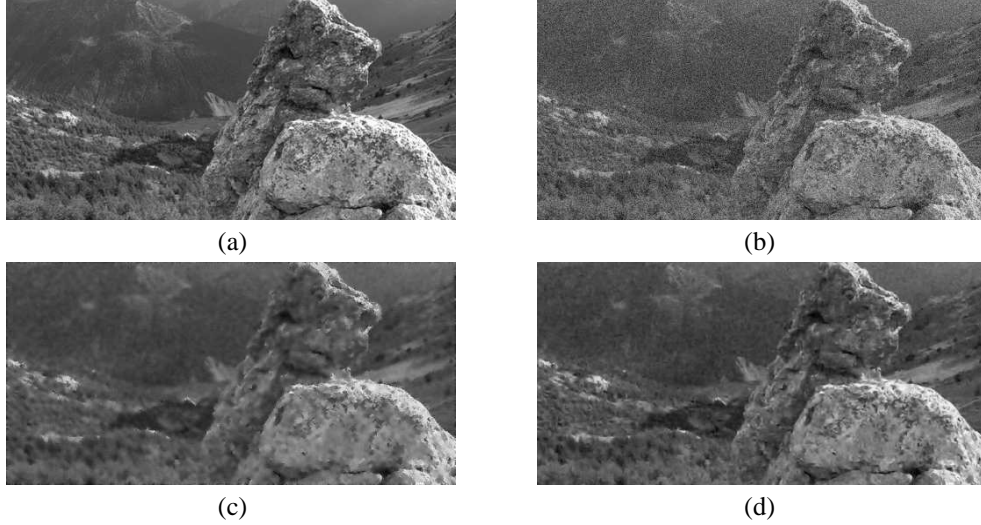

(a)                                                          (b)

(c)                                                          (d)

Figure 3: (a)- Original 'rock'-image. (b)-Rock-image with noise added. (c)-Denoised image using Wiener filtering. (d) Denoised image using IWF.

To minimize this we follow a variational procedure where we upper bound the logarithm using $\log x \le \gamma x - \log \gamma - 1$. The bound is saturated when $\gamma = 1/x$. Applying this to every logarithm in the summation in eqn. (10) and iteratively minimizing this bound over $\mathbf{x}$ and $\boldsymbol{\gamma}$ we find the following update equations,

$$1/\gamma_i \quad \leftarrow \quad 1 + \frac{1}{2}\sum_j W_{ij}(\mathbf{J}_j^T\mathbf{x})^2 \tag{11}$$

$$\mathbf{x}^{\mathbf{MAP}} \quad \leftarrow \quad (\boldsymbol{\Sigma}^{-1} + \mathbf{JDJ}^T)^{-1}\boldsymbol{\Sigma}^{-1}\mathbf{y} \qquad \mathbf{D} = \mathbf{Diag}[\mathbf{W}^T(\boldsymbol{\alpha}\times\boldsymbol{\gamma})] \tag{12}$$

where $\times$ denotes componentwise multiplication. Since the second equation is just a Wiener filter with noise covariance $\boldsymbol{\Sigma}$ and a Gaussian prior with covariance $(\mathbf{JDJ}^T)^{-1}$ we have named the above denoising equations the iterated Wiener filter (IWF).

When the filters are orthonormal, the noise covariance isotropic and the weight matrix the identity, the minimization in (10) decouples into $M$ minimizations over the transformed variables $z_i = \mathbf{J}_i^T\mathbf{x}$. Defining $v_i = \mathbf{J}_i^T\mathbf{y}$ we can easily derive that $z_i$ is the solution of the following cubic equation (for which analytic solutions exist),

$$z_i^3 - v_i z_i^2 + 2(1 + \sigma^2\alpha_i)z_i - 2v_i = 0 \tag{13}$$

We note however that constraining the filters to be orthogonal is a rather severe restriction if the data are not pre-whitened. On the other hand, if we decide to work with whitened data, the isotropic noise assumption seems unrealistic. Having said that, Hyvarinen's shrinkage method for ICA models [3] is based on precisely these assumptions and seems to give good results. The proposed method is also related to approaches based on the GSM [7].

## 6.1   Experiment: Denoising

To test the iterated Wiener filter, we trained a complete set of 99 experts on the data described in section (4.1). The norm of the filters was unconstrained, the $\boldsymbol{\alpha}$ were free to adapt, but we did not include any weights $\mathbf{W}$. The image shown in figure (3a) was corrupted with Gaussian noise with standard deviation $\sigma = 20$, which resulted in a PSNR of 22.0dB (figure (3b)). We applied the adaptive Wiener filter from matlab (Wiener2.m) with an optimal

$5 \times 5$ neighborhood size and known noise-variance. The denoised image using adaptive Wiener filtering has a PSNR of 26.9dB and is shown in figure (3c). IWF was run on every possible $10 \times 10$ patch in the image, after which the results were averaged. Because the filters $\mathbf{J}^T$ were trained on sphered data without a DC component, the same transformations have to be applied to the test patches before IWF is applied. The denoised image using IWF is shown in (3d) and has a PSNR of 28.1dB, which is a significant improvement of 1.2dB over Wiener filtering. It is our hope that the use of overcomplete representations and weights $\mathbf{W}$ will further improve those results.

## 7 Discussion

It is well known that a wavelet transform de-correlates natural image data in good approximation. In [6] it was found that in the marginal distribution the wavelet coefficients are sparsely distributed but that there are significant residual dependencies among their energies $z_i^2$. In this paper we have shown that the PoT model can learn highly overcomplete filters with sparsely distributed outputs. With a second hidden layer that is locally connected, it captures the dependencies between filter outputs by learning topographic representations.

Our approach improves upon earlier attempts (e.g. [4],[8]) in a number of ways. In the PoT model the hidden variables are *conditionally* independent so perceptual inference is very easy and does not require iterative settling even when the model is overcomplete. There is a fairly simple and efficient procedure for learning all the parameters, including the weights connecting top-level units to filter outputs. Finally, the model leads to an elegant denoising algorithm which involves iterating a Wiener-filter.

### Acknowledgements

This research was funded by NSERC, the Gatsby Charitable Foundation, and the Wellcome Trust. We thank Yee-Whye Teh for first suggesting a related model and Peter Dayan for encouraging us to apply products of experts to topography.

## Footnotes

[1]Collected from http://www.cis.hut.fi /projects/ica/data/images

[2]Interestingly, the update equations for the filters $\mathbf{J}$ presented in [4], which minimize a bound on the log-likelihood of a directed model, reduce to the same equations as our learning rules when the representation is complete and the filters orthogonal.

## References

[1] G.E. Hinton. Training products of experts by minimizing contrastive divergence. *Neural Computation*, 14:1771–1800, 2002.

[2] G.E. Hinton, M. Welling, Y.W. Teh, and K. Osindero. A new view of ICA. In *Int. Conf. on Independent Component Analysis and Blind Source Separation*, 2001.

[3] A. Hyvarinen. Sparse code shrinkage: Denoising of nongaussian data by maximum likelihood estimation. *Neural Computation*, 11(7):1739–1768, 1999.

[4] A. Hyvarinen, P.O. Hoyer, and M. Inki. Topographic independent component analysis. *Neural Computation*, 13(7):1525–1558, 2001.

[5] S. Della Pietra, V.J. Della Pietra, and J.D. Lafferty. Inducing features of random fields. *IEEE Transactions on Pattern Analysis and Machine Intelligence*, 19(4):380–393, 1997.

[6] E.P. Simoncelli. Modeling the joint statistics of images in the wavelet domain. In *Proc SPIE, 44th Annual Meeting*, volume 3813, pages 188–195, Denver, 1999.

[7] V. Strela, J. Portilla, and E. Simoncelli. Image denoising using a local Gaussian scale mixture model in the wavelet domain. In *Proc. SPIE, 45th Annual Meeting*, San Diego, 2000.

[8] M.J. Wainwright and E.P. Simoncelli. Scale mixtures of Gaussians and the statistics of natural images. In *Advances Neural Information Processing Systems*, volume 12, pages 855–861, 2000.

[9] S.C. Zhu, Z.N. Wu, and D. Mumford. Minimax entropy principle and its application to texture modeling. *Neural Computation*, 9(8):1627–1660, 1997.
